# An analysis on negative curvature induced by singularity in multi-layer neural-network learning

**Eiji Mizutani**
Department of Industrial Management
Taiwan Univ. of Science & Technology
`eiji@mail.ntust.edu.tw`

**Stuart Dreyfus**
Industrial Engineering & Operations Research
University of California, Berkeley
`dreyfus@ieor.berkeley.edu`

## Abstract

In the neural-network parameter space, an attractive field is likely to be induced by singularities. In such a singularity region, first-order gradient learning typically causes a long plateau with very little change in the objective function value $E$ (hence, a flat region). Therefore, it may be confused with "attractive" local minima. Our analysis shows that the Hessian matrix of $E$ tends to be *indefinite* in the vicinity of (perturbed) singular points, suggesting a promising strategy that exploits negative curvature so as to escape from the singularity plateaus. For numerical evidence, we limit the scope to small examples (some of which are found in journal papers) that allow us to confirm singularities and the eigenvalues of the Hessian matrix, and for which computation using a descent direction of negative curvature encounters no plateau. Even for those small problems, no efficient methods have been previously developed that avoided plateaus.

## 1   Introduction

Consider a general two-hidden-layer multilayer perceptron (MLP) having a single (terminal) output, $H$ nodes at the second hidden layer (next to the terminal layer), $I$ nodes at the first hidden layer, and $J$ nodes at the input layer; hence, a $J$-$I$-$H$-1 MLP. It has totally $n$ parameters, denoted by an $n$-vector $\boldsymbol{\theta}$, including thresholds. Let $\phi(.)$ be some node function; then, the forward pass transforms the input vector $\mathbf{x}$ of length $J$ to the first hidden-output vector $\mathbf{z}$ of length $I$, and then to the second hidden-output vector $\mathbf{h}$ of length $H$, leading to the final output $y$:

$$y = f(\boldsymbol{\theta}; \mathbf{x}) = \phi\left(\mathbf{h}_+^T \mathbf{p}\right) = \phi\left(\sum_{j=0}^{H} p_j h_j\right) = \phi\left(\sum_{j=0}^{H} p_j \phi(\mathbf{z}_+^T \mathbf{v}_j)\right) \text{ with } z_k = \phi(\mathbf{x}_+^T \mathbf{w}_k). \tag{1}$$

Here, fictitious outputs $x_0 = z_0 = h_0 = 1$ are included in the output vectors with subscript "+" for thresholds $p_0$, $v_{0,j}$, and $w_{0,k}$; $p_j$ ($j = 1, ..., H$) is the weight connecting the $j$th hidden node to the (final) output; $\mathbf{v}_j$ a vector of "hidden" weights directly connecting to the $j$th hidden node from the first hidden layer; $\mathbf{w}_k$ a vector of "hidden" weights to the $k$th hidden node from the input layer; hence, $\boldsymbol{\theta}^T \equiv [\mathbf{p}^T | \mathbf{v}^T | \mathbf{w}^T] = [\mathbf{p}^T | \mathbf{v}_1^T, ..., \mathbf{v}_j^T, ..., \mathbf{v}_H^T | \mathbf{w}_1^T, ..., \mathbf{w}_k^T, ..., \mathbf{w}_I^T]$. The length of those weight vectors $\boldsymbol{\theta}$, $\mathbf{p}$, $\mathbf{v}$, $\mathbf{w}$ are denoted by $n$, $n_3$, $n_2$, and $n_1$, respectively, where

$$n = n_3 + n_2 + n_1; \quad n_3 = (H+1); \quad n_2 = H(I+1); \quad n_1 = I(J+1). \tag{2}$$

For parameter optimization, one may attempt to minimize the squared error over $m$ data

$$E(\boldsymbol{\theta}) = \frac{1}{2}\sum_{d=1}^{m}\{f(\boldsymbol{\theta}; \mathbf{x}_d) - t_d\}^2 = \frac{1}{2}\sum_{d=1}^{m} r_d^2(\boldsymbol{\theta}) = \frac{1}{2}\mathbf{r}^T\mathbf{r}, \tag{3}$$

where $t_d$ is a desired output on datum $d$; each residual $r_d$ a smooth function from $\Re^n$ to $\Re$; and $\mathbf{r}$ an $m$-vector of residuals. Note here and hereafter that the argument $(\boldsymbol{\theta})$ for $E$ and $\mathbf{r}$ is frequently suppressed as long as no confusion arises. The gradient and Hessian of $E$ can be expressed as below

$$\nabla E(\boldsymbol{\theta}) = \sum_{d=1}^{m} r_d \nabla r_d = \mathbf{J}^T\mathbf{r}, \text{ and } \nabla^2 E(\boldsymbol{\theta}) = \sum_{d=1}^{m}\nabla r_d \nabla r_d^T + \sum_{d=1}^{m} r_d \nabla^2 r_d \equiv \mathbf{J}^T\mathbf{J} + \mathbf{S}, \tag{4}$$

where $\mathbf{J} \equiv \nabla\mathbf{r}$, an $m \times n$ Jacobian matrix of $\mathbf{r}$, and the $d$th row of $\mathbf{J}$ is denoted by $\nabla r_d^T$.

In the well-known Gauss-Newton method, $\mathbf{S}$, the last matrix of second derivatives of residuals, is omitted, and its search direction $\Delta\boldsymbol{\theta}$ is found by solving $\mathbf{J}\Delta\boldsymbol{\theta}_{\text{GN}} = -\mathbf{r}$ (or, $\mathbf{J}^T\mathbf{J}\Delta\boldsymbol{\theta}_{\text{GN}} = -\nabla E$). Under the normal error assumption, the Fisher information matrix is tantamount to $\mathbf{J}^T\mathbf{J}$, called the Gauss-Newton Hessian. This is why natural gradient learning can be viewed as an incremental version of the Gauss-Newton method (see p.1404 [1]; p.1031 [2]) in the nonlinear least squares sense. Since $\mathbf{J}^T\mathbf{J}$ is positive (semi)definite, natural gradient learning has no chance to exploit negative curvature. It would be of great value to understand the weaknesses of such Gauss-Newton-type methods.

Learning behaviors of layered networks may be attributable to *singularities* [3, 2, 4]. Singularities have been well discussed in the nonlinear least squares literature also: For instance, Jennrich & Sampson (pp.65–66 [5]) described an overlap-singularity situation involving a *redundant* model; specifically, a classical (linear-output) model of exponentials with $h_i \equiv \phi(v_i x)$ and no thresholds in Eq.(1):

$$f(\boldsymbol{\theta}; x) = p_1\phi(v_1 x) + p_2\phi(v_2 x) = p_1 e^{v_1 x} + p_2 e^{v_2 x}. \tag{5}$$

If the target data follow the path of a single exponential then the two hidden parameters, $v_1$ and $v_2$, become identical (i.e., overlap singularity) at the solution point, where $\mathbf{J}$ is *rank-deficient*; hence, $\mathbf{J}^T\mathbf{J}$ is *singular*. If the fitted response function nearly follows such a path, then $\mathbf{J}^T\mathbf{J}$ is nearly singular. This is a typical over-realizable scenario, in which the true teacher lies at the singularity (see [6] for details about 1-2-1 MLP-learning). In practice, if the solution point $\boldsymbol{\theta}_*$ is stationary but $\mathbf{J}(\boldsymbol{\theta}_*)$ is rank-deficient, then the search direction $\Delta\boldsymbol{\theta}_{\text{GN}}$ can be numerically orthogonal to $\nabla E$ at some distant point from $\boldsymbol{\theta}_*$; consequently, no progress can be made by searching along the Gauss-Newton direction (hence, line-search-based algorithms fail); this is first pointed out by Powell, who proved in [7] that the Gauss-Newton iterates converge to a non-stationary limit point at which $\mathbf{J}$ is rank-deficient in solving a particular system of nonlinear equations, for which the merit function is defined as Eq.(3), where $m = n$. Another weak point of the Gauss-Newton-type method is a so-called *large-residual* problem (e.g., see Dennis [8]); this implies that $\mathbf{S}$ in $\nabla^2 E$ is substantial because $\mathbf{r}$ is highly nonlinear, or its norm is large at solution $\boldsymbol{\theta}_*$. Those drawbacks of the Gauss-Newton-type methods indicate that negative curvature often arises in MLP-learning when $\mathbf{J}^T\mathbf{J}$ is *singular* (i.e., in a *rank-deficient* nonlinear least squares problem), and/or when $\mathbf{S}$ is more dominant than $\mathbf{J}^T\mathbf{J}$. We thus verify this fact mathematically, and then discuss how exploiting negative curvature is a good way to escape from singularity plateaus, thereby enhancing the learning capacity.

## 2 Negative curvature induced by singularity

In rank-deficient nonlinear least squares problems, where $\mathbf{J} \equiv \nabla\mathbf{r}$ is rank deficient, negative curvature often arises. This is true with an *arbitrary* MLP model, but to make our analysis concrete, we consider a single terminal linear-output two-hidden-layer MLP: $f(\boldsymbol{\theta}; \mathbf{x}) = \sum_{j=0}^{H} p_j h_j$ in Eq. (1). Then, the $n$ weights separate into linear $\mathbf{p}$ and non-linear $\mathbf{v}$ and $\mathbf{w}$. In this context, we show that a 4-by-4 *indefinite* Hessian block can be extracted from the $n$-by-$n$ Hessian matrix $\nabla^2 E$ in Eq.(4).

### 2.1 An existence of the $4 \times 4$ indefinite Hessian block H in $\nabla^2 E$

In the posed two-hidden-layer MLP-learning, as indicated after Eq.(1), the $n$ weights are organized as $\boldsymbol{\theta}^T \equiv [\mathbf{p}^T | \mathbf{v}^T | \mathbf{w}^T]$. Now, we pay attention to two particular hidden nodes $j$ and $k$ at the second hidden layer. The weights connecting to those two nodes are $p_j$, $p_k$, $\mathbf{v}_j$, and $\mathbf{v}_k$; they are arranged in the following manner:

$$\boldsymbol{\theta}^T = [p_0, p_1, ..., p_j, ..., p_k, ..., p_H | v_{0,1}, ........, | v_{0,j}, v_{1,j}, ..., v_{I,j} | ... | v_{0,k}, v_{1,k}, ..., v_{I,k} | .... , | \quad \mathbf{w}^T], \tag{6}$$

where $v_{i,k}$ is a weight from node $i$ at the first hidden layer to node $k$ at the second hidden layer. Given a data pair $(\mathbf{x}; t)$, $r \equiv f(\boldsymbol{\theta}; \mathbf{x}) - t$, a residual element, and $\mathbf{u}^T$, an $n$-length row vector of the residual Jacobian matrix $\mathbf{J}$ ($\equiv \frac{\partial r}{\partial \boldsymbol{\theta}}$) in Eq.(4), is given as below using the output vector $\mathbf{z}_+$ (including $z_0 = 1$) at the first hidden layer

$$\mathbf{u}^T \equiv \nabla r^T = [..., h_j, ..., h_k, ..., \phi_j'(\mathbf{z}_+^T \mathbf{v}_j)p_j, ..., \phi_k'(\mathbf{z}_+^T \mathbf{v}_k)p_k, ...], \tag{7}$$

where only four entries are shown that are associated with four weights: $p_j$, $p_k$, $v_{0,j}$, and $v_{0,k}$. The locations of those four weights in the $n$-vector $\boldsymbol{\theta}$ are denoted by $l_1$, $l_2$, $l_3$, and $l_4$, respectively, where

$$l_1 \equiv j+1, \quad l_2 \equiv k+1, \quad l_3 \equiv (I+1)(j-1)+1, \quad l_4 \equiv (I+1)(k-1)+1. \tag{8}$$

Given $\mathbf{J}$, we interchange columns 1 and $l_1$; then, do columns 2 and $l_2$; then columns 3 and $l_3$; and finally columns 4 and $l_4$; this interchanging procedure moves those four columns to the first four.

Suppose that the $n \times n$ Hessian matrix $\nabla^2 E = \mathbf{u}\mathbf{u}^T + \mathbf{S}$ is evaluated on a given single datum $(\mathbf{x}; t)$. We then apply the above interchanging procedure to both rows and columns of $\nabla^2 E$ appropriately, which can be readily accomplished by $\mathbf{P}^T \nabla^2 E \mathbf{P}$, where four permutation matrices $\mathbf{P}_i$ $(i = 1, ..., 4)$ are employed as $\mathbf{P} \equiv \mathbf{P}_1 \mathbf{P}_2 \mathbf{P}_3 \mathbf{P}_4$; each $\mathbf{P}_i$ satisfies $\mathbf{P}_i^T \mathbf{P}_i = \mathbf{I}$ (orthogonal) and $\mathbf{P}_i = \mathbf{P}_i^T$ (symmetric); hence, $\mathbf{P}$ is *orthogonal*. As a result, $\mathbf{H}$, the 4-by-4 Hessian block (at the upper-left corner) of the first four leading rows and columns of $\mathbf{P}^T \nabla^2 E \mathbf{P}$ has the following structure:

$$\underbrace{\mathbf{H}}_{4 \times 4} = \begin{bmatrix} (h_j)^2 & h_j h_k & h_j \phi'_j(.) p_j & h_j \phi'_k(.) p_k \\ & (h_k)^2 & h_k \phi'_j(.) p_j & h_k \phi'_k(.) p_k \\ & & \{\phi'_j(.) p_j\}^2 & \phi'_j(.) \phi'_k(.) p_j p_k \\ & \text{Symmetric} & & \{\phi'_k(.) p_k\}^2 \end{bmatrix} + \begin{bmatrix} 0 & 0 & \phi'_j(.) r & 0 \\ & 0 & 0 & \phi'_k(.) r \\ & & \phi''_j(.) p_j r & 0 \\ & \text{Symmetric} & & \phi''_k(.) p_k r \end{bmatrix}. \quad (9)$$

The posed Hessian block $\mathbf{H}$ is associated with a vector of the four weights $[p_j, p_k, v_{0,j}, v_{0,k}]^T$.

If $\mathbf{v}_j = \mathbf{v}_k$, then $h_j = h_k = \phi(\mathbf{z}_+^T \mathbf{v})$; see Eq.(7). Obviously, no matter how many data are accumulated, two columns $\mathbf{h}_j$ and $\mathbf{h}_k$ of $\mathbf{J}$ in Eq.(4) are identical; therefore, $\mathbf{J}$ is rank deficient; hence, $\mathbf{J}^T \mathbf{J}$ is *singular*. The posed singularity gives rise to negative curvature because the above 4-by-4 *dense* Hessian block is almost always *indefinite* (so is $\nabla^2 E$ of size $n \times n$) to be proved next.

## 2.2  Case 1: $\mathbf{v}_j = \mathbf{v}_k \equiv \mathbf{v}$; hence, $h_j = h_k \equiv h = \phi(\mathbf{z}_+^T \mathbf{v})$, and $p_j \neq p_k$

Given a set of $m$ (training) data, the gradient vector $\nabla E$ and the Hessian matrix $\nabla^2 E$ in Eq.(4) are evaluated. We then apply the aforementioned *orthogonal* matrix $\mathbf{P}$ to them as $\mathbf{P}^T \nabla E$ and $\mathbf{P}^T \nabla^2 E \mathbf{P}$, yielding the gradient vector $\mathbf{g}$ of length 4 and the 4-by-4 Hessian block $\mathbf{H}$ [see Eq.(9)] associated with the four weights $[p_j, p_k, v_{0,j}, v_{0,k}]^T$; they may be expressed in a compact form as

$$\mathbf{g} = \sum_{d=1}^{m} r_d \mathbf{u}_d = \begin{bmatrix} \gamma \\ \gamma \\ p_j e \\ p_k e \end{bmatrix}; \quad \mathbf{H} = \mathbf{J}^T \mathbf{J} + \mathbf{S} = \begin{bmatrix} a & a & b_1 & b_2 \\ a & a & b_1 & b_2 \\ b_1 & b_1 & c_{11} & c_{12} \\ b_2 & b_2 & c_{12} & c_{22} \end{bmatrix} + \begin{bmatrix} 0 & 0 & e & 0 \\ 0 & 0 & 0 & e \\ e & 0 & d_1 & 0 \\ 0 & e & 0 & d_2 \end{bmatrix}, \quad (10)$$

where the entries are given below with $B \equiv \sum_{d=1}^{m} \phi'(\mathbf{z}_{+_d}^T \mathbf{v}) h_d$, $C \equiv \sum_{d=1}^{m} \left(\phi'(\mathbf{z}_{+_d}^T \mathbf{v})\right)^2$, $D \equiv \sum_{d=1}^{m} \phi''(\mathbf{z}_{+_d}^T \mathbf{v}) r_d$:

$$\begin{cases} a \equiv \sum_{d=1}^{m} h_d^2, & b_1 \equiv p_j B, \quad b_2 \equiv p_k B, \quad c_{11} \equiv p_j^2 C, \quad c_{12} \equiv p_j p_k C, \quad c_{22} \equiv p_k^2 C, \\ \gamma \equiv \sum_{d=1}^{m} r_d h_d, & e \equiv \sum_{d=1}^{m} \phi'(\mathbf{z}_{+_d}^T \mathbf{v}) r_d, \quad d_1 \equiv p_j D, \quad d_2 \equiv p_k D. \end{cases} \quad (11)$$

Notice here that the subscript $d$ implies datum $d$ $(d = 1, ..., m)$; hence, $h_d$ is the hidden-node output on datum $d$ (but not the $d$th hidden-node output) common to both nodes $j$ and $k$ due to $\mathbf{v}_j = \mathbf{v}_k = \mathbf{v}$.

**Theorem 1**: *When $e \neq 0$, the $n$-by-$n$ Hessian $\nabla^2 E$ and its block $\mathbf{H}$ in Eq.(10) are always indefinite.*
**Proof:** A *similarity transformation* with $\mathbf{T}$, a 4-by-4 *orthogonal* matrix $(\mathbf{T}^T = \mathbf{T}^{-1})$, obtains

$$\mathbf{T}^T \mathbf{H} \mathbf{T} = \begin{bmatrix} 2a & b_1 + b_2 + e & 0 & b_1 - b_2 \\ b_1 + b_2 + e & \alpha & 0 & \beta \\ 0 & 0 & 0 & e \\ b_1 - b_2 & \beta & e & \tau \end{bmatrix} \quad \text{with } \mathbf{T} = \begin{bmatrix} \frac{1}{\sqrt{2}} & 0 & \frac{1}{\sqrt{2}} & 0 \\ \frac{1}{\sqrt{2}} & 0 & \frac{-1}{\sqrt{2}} & 0 \\ 0 & \frac{1}{\sqrt{2}} & 0 & \frac{1}{\sqrt{2}} \\ 0 & \frac{1}{\sqrt{2}} & 0 & \frac{-1}{\sqrt{2}} \end{bmatrix}, \quad (12)$$

where $\alpha \equiv \frac{1}{2}(c_{11} + 2c_{12} + c_{22} + d_1 + d_2)$, $\beta \equiv \frac{1}{2}(c_{11} - c_{22} + d_1 - d_2)$, and $\tau \equiv \frac{1}{2}(c_{11} - 2c_{12} + c_{22} + d_1 + d_2)$. The eigenvalues of the 2-by-2 block at the lower-right corner are obtainable by

$$\left| \lambda \mathbf{I} - \begin{bmatrix} 0 & e \\ e & \tau \end{bmatrix} \right| = \lambda(\lambda - \tau) - e^2 = \lambda^2 - \tau\lambda - e^2 = 0,$$

which yields $\frac{1}{2}(\tau \pm \sqrt{\tau^2 + 4e^2})$, the "sign-different" eigenvalues as long as $e \neq 0$ holds. Then, by Cauchy's interlace theorem (see Ch.10 of Parlett 1998), the Hessian $\nabla^2 E$ is indefinite. (So is $\mathbf{H}$.) □

## 2.3  Case 2: $\mathbf{v}_j = \mathbf{v}_k \equiv \mathbf{v}$ $(h_j = h_k \equiv h)$, and $p_j = p_k \equiv p$

The result in Case 1 becomes simpler: For a given set of $m$ (training) data,

$$\mathbf{g} = \sum_{d=1}^{m} r_d \mathbf{u}_d = \begin{bmatrix} \gamma \\ \gamma \\ pe \\ pe \end{bmatrix}; \quad \mathbf{H} = \mathbf{J}^T \mathbf{J} + \mathbf{S} = \begin{bmatrix} a & a & b & b \\ a & a & b & b \\ b & b & c & c \\ b & b & c & c \end{bmatrix} + \begin{bmatrix} 0 & 0 & e & 0 \\ 0 & 0 & 0 & e \\ e & 0 & d & 0 \\ 0 & e & 0 & d \end{bmatrix}, \quad (13)$$

where the entries are readily identifiable from Eq.(11). In Eq.(13), $\mathbf{J}^T\mathbf{J}$ is positive semi-definite (*singular* of rank 2 even when $m \geq 2$), and $\mathbf{S}$ has an *indefinite* structure. When $e \neq 0$ (hence, $\nabla E \neq \mathbf{0}$), we can prove below that there always exists negative curvature (i.e., $\nabla^2 E$ is always *indefinite*).

**Theorem 2**: *When $e \neq 0$, the $4 \times 4$ Hessian block $\mathbf{H}$ in Eq.(13) includes the sign-different eigenvalues of $\mathbf{S}$; namely, $\frac{1}{2}(d \pm \sqrt{d^2 + 4e^2})$, and the $n \times n$ Hessian $\nabla^2 E$ as well as $\mathbf{H}$ are always indefinite.*
**Proof:** Proceed similarly with the same orthogonal matrix $\mathbf{T}$ as defined in Eq.(12), where $b_1 = b_2 = b$, $\beta = 0$, and $\tau = d$, rendering $\mathbf{T}^T\mathbf{HT}$ "block-diagonal." Its block of size $2 \times 2$ at the lower-right corner has the sign-different eigenvalues determined by $\lambda^2 - d\lambda - e^2 = 0$. □ *QED* □

Now, we investigate *stationary points*, where the $n$-length gradient vector $\nabla E = \mathbf{0}$; hence, $\mathbf{g} = \mathbf{0}$ in Eq.(13). We thus consider two cases for $pe = 0$: **(a)** $p = 0$ and $e \neq 0$, and **(b)** $p \neq 0$ and $e = 0$. In Case **(b)**, $\mathbf{S}$ becomes a diagonal matrix, and the above $\mathbf{T}^T\mathbf{HT}$ shows that $\mathbf{H}$ is of (at most) rank 3 (when $d \neq 0$); hence, $\mathbf{H}$ becomes *singular*.

**Theorem 3**: *If $\nabla E(\boldsymbol{\theta}^*) = \mathbf{0}$, $p = 0$, and $e \neq 0$ [i.e., Case (a)], then the stationary point $\boldsymbol{\theta}^*$ is a saddle.*
**Theorem 4**: *If $\nabla E(\boldsymbol{\theta}^*) = \mathbf{0}$, and $e = 0$, but $d < 0$ [see Eq.(13)], then $\boldsymbol{\theta}^*$ is a saddle point.*
**Proof of Theorems 3 and 4**: From Theorem 2 above, $\mathbf{H}$ in Eq.(13) has a negative eigenvalue; hence, the entire Hessian matrix $\nabla^2 E$ of size $n \times n$ is indefinite □ *QED* □

Theorem 4 is a special case of Case **(b)**. If $d = pD > 0$, then $\mathbf{H}$ becomes positive semi-definite; however, we could alter the eigen-spectrum of $\mathbf{H}$ by changing linear parameters $p$ in conjunction with scalar $\zeta$ for $p_j = 2\zeta p$ and $p_k = 2(1-\zeta)p$ such that $p_j + p_k = 2p$ with no change in $E$ and $\nabla E = \mathbf{0}$ held fixed (to be confirmed in simulation; see Fig.1 later), leading to the following

**Theorem 5**: *If $D \neq 0$ and $C > 0$ [see the definition of $C$ and $D$ for Eq.(11)] and $v_1 = v_2 (\equiv v)$ with $\nabla E = \mathbf{0}$, for which $p \neq 0$ and $e = 0$ (hence, $\mathbf{S}$ is diagonal), then choosing scalar $\zeta$ appropriately for $p_j = 2\zeta p$ and $p_k = 2(1-\zeta)p$ can render $\mathbf{H}$ and thus $\nabla^2 E$ indefinite.*
**Proof:** From Eq.(11), two on-diagonal (3,3) and (4,4) entries of $\mathbf{H}$ are a *quadratic* function in terms of $\zeta$: The (3,3)-entry of $\mathbf{H}$, $\mathbf{H}(3,3) = 2\zeta p(2\zeta pC + D)$, has two roots: 0 and $-\frac{D}{2pC}$, whereas the (4,4)-entry, $\mathbf{H}(4,4) = 2(1-\zeta)p[2(1-\zeta)pC+D]$, has two roots: 1 and $1 + \frac{D}{2pC}$. Obviously, given $p$, $C$, and $D$, there exists $\zeta$ such that the quadratic function value becomes *negative* (see later Fig.1). This implies that adjusting $\zeta$ can produce a negative diagonal entry of $\mathbf{H}$; hence, *indefinite*. Then, again by Cauchy's interlace theorem, so is $\nabla^2 E$. □ *QED* □

**Example 1**: A two-exponential model in Eq.(5).

**Data set 1**:

| Input $x$ | $-2$ | $-1$ | 0 | 1 | 2 |
|---|---|---|---|---|---|
| Target $t$ | 1 | 3 | 2 | 3 | 1 |

**Data set 2**:

| Input $x$ | $-2$ | $-1$ | 0 | 1 | 2 |
|---|---|---|---|---|---|
| Target $t$ | 3 | 1 | 2 | 1 | 3 |

(14)

Given two sets of five data pairs $(x_i; t_i)$ as shown above, for each data set, we first find a minimizer $\boldsymbol{\theta}^0_* = [p_*, v_*]^T$ of a two-weight 1-1-1 MLP, and then expand it with scalar $\zeta$ as $\boldsymbol{\theta} = [\zeta p_*, (1-\zeta)p_*, v_*, v_*]^T$ to construct a four-weight 1-2-1 MLP that produces the same input-to-output relations. That is, we first find the minimizer $\boldsymbol{\theta}^0_* = [p_*, v_*]^T$ using a 1-1-1 MLP, $f(\boldsymbol{\theta}^0; x) = pe^{vx}$, by solving $\nabla E = \mathbf{0}$, which yields $p_* = 2$; $v_* = 0$; $E(\boldsymbol{\theta}^0_*) = \frac{1}{2}\sum_{j=1}^5 \{f(\boldsymbol{\theta}^0; x_j) - t_j\}^2 = 2$; and confirm that the $2 \times 2$ Hessian $\nabla^2 E(\boldsymbol{\theta}^0_*)$ is positive definite in both data sets above. Next, we augment $\boldsymbol{\theta}^0_*$ as $\boldsymbol{\theta} = [p_1, p_2, v_1, v_2]^T = [\zeta p_*, (1-\zeta)p_*, v_*, v_*]^T$ to construct a 1-2-1 MLP: $f(\boldsymbol{\theta}; x) = p_1 e^{v_1 x} + p_2 e^{v_2 x}$, which realizes the same input-to-output relations as the 1-1-1 MLP. Fig.1 shows how $\zeta$ changes the eigen-spectrum (see solid curve) of the $4 \times 4$ Hessian $\nabla^2 E$ (supported by Theorem 5).

**Conjecture**: *Suppose that $\boldsymbol{\theta}^*$ is a local minimum point in two-hidden-layer J-I-H-1 MLP-learning, and $\nabla^2 E$ of size $n \times n$ is positive definite (so is $\mathbf{H}$) with $\nabla E = \mathbf{0}$ and $E > 0$. Then, adding a node at the second hidden layer can increase learning capacity in the sense that $E$ can be further reduced.*
**Sketch of Proof:** Choose a node $j$ among $H$ hidden nodes, and add a hidden node (call node $k$) by duplicating the hidden weights by $\mathbf{v}_k = \mathbf{v}_j$ with $p_k = 0$; hence, totally $\widetilde{n} \equiv n + (I+2)$ weights. This certainly renders new $\mathbf{J}^T\mathbf{J}$ of size $\widetilde{n} \times \widetilde{n}$ *singular*, and the (4,4)-entry in $\mathbf{H}$ in Eq.(10) becomes zero (due to $p_k = 0$). Then, by the interlace theorem, new $\nabla^2 E$ of size $\widetilde{n} \times \widetilde{n}$ becomes indefinite. □

The above proof is not complete since we did not make clear assumptions about how the first-order necessary condition $\nabla E = \mathbf{0}$ holds [see **Cases (a)** and **(b)** just above Theorem 3]. Furthermore, even if we know in advance the minimum number of hidden nodes, $H_{\min}$, for a certain task, we may not be able to find a local-minimum point of an MLP with one less hidden nodes, $H_{\min} - 1$. Consider, for instance, the well-known (four data) XOR problem. Although it can be solved by a 2-2-1 MLP (nine

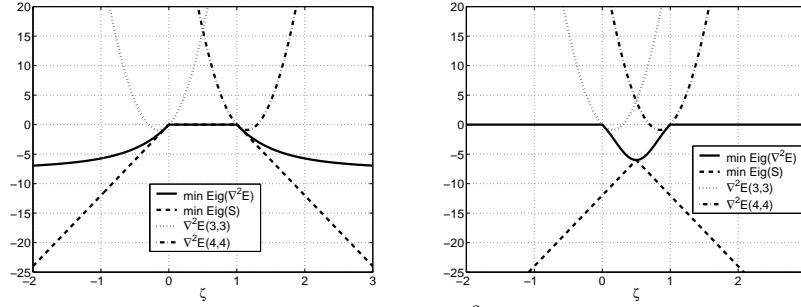

Figure 1: The change of the minimum eigenvalue of $\nabla^2 E$ (*solid* curve) and of **S** (dashed) as well as the (3,3)-entry of $\nabla^2 E$ (dotted) and the (4,4)-entry of $\nabla^2 E$ (dash-dot), both quadratic, according to value $\zeta$ (x-axis) in $\boldsymbol{\theta} = [\zeta p_*, (1-\zeta)p_*, v_*, v_*]^T$, the four weights of a 1-2-1 MLP with exponential hidden nodes (left) using data set 1, and (right) data set 2 in Eq.(14). Theorem 5 supports this result.

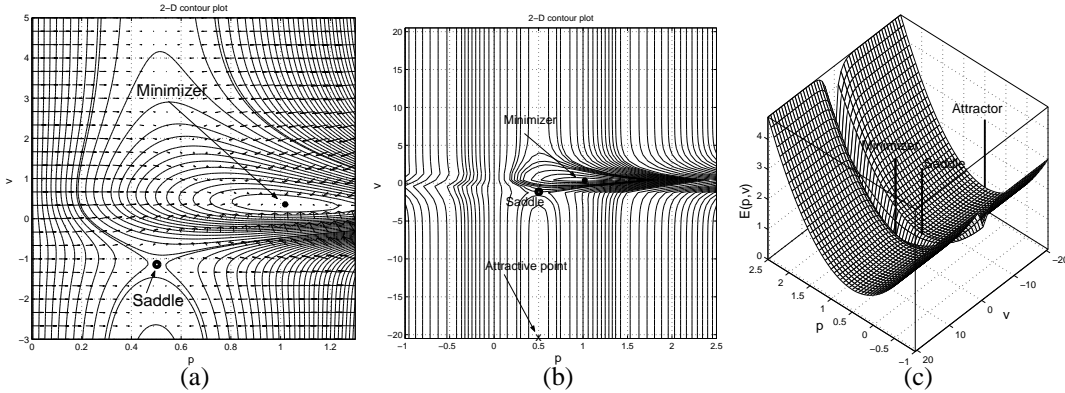

(a)                 (b)                 (c)

Figure 2: The 1-1-1 MLP landscape: (a) a magnified view; (b) bird's-eye views in 2-D, and (c) 3-D.

weights), any local minimum point may not be found by optimizing a 2-1-1 MLP (five weights), since the hidden weights tend to be divergent (or weight-$\infty$ attractors). Here is another example:

**Example 2:** An $N$-shape curve fitting to four data: $\mathsf{Data}(\mathbf{x}; \mathbf{t}) \equiv \{(-3; 0), (-1; 1), (1; 0), (3; 1)\}$. We solved $\nabla E = \mathbf{0}$ to find all stationary points of a two-weight 1-1-1 MLP with a logistic hidden-node function $\phi(x) \equiv \frac{1}{1+e^{-x}}$, and found $p_* \approx 1.0185$ and $v_* \approx 0.3571$ with $\|\nabla E(\boldsymbol{\theta}_*^0)\| = O(10^{-15})$, roughly the order of machine (double) precision, and $E(\boldsymbol{\theta}_*^0) \approx 0.4111$. The Hessian $\nabla^2 E(\boldsymbol{\theta}_*^0)$ was positive definite (eigenvalues: 0.8254 and 1.4824). We also found a saddle point. There was another type of *attractive* points, where $\phi$ is driven to saturation due to a large hidden weight $v$ in magnitude (weight-$\infty$ attractors). Fig.2 displays those three types of stationary points. Clearly, for a rigorous proof of Conjecture, we need to characterize those different types, and clarify their underlying assumptions; yet, it is quite an arduous task because the situation totally depends on data; see also our Hessian argument for Blum's line in Sec.3.2.

We continue with **Example 2** to verify the above theorems. We set $\boldsymbol{\theta} = [\zeta p_*, (1-\zeta)p_*, v_*, v_*]$ in a 1-2-1 MLP. When $\zeta = 0.5$, the Hessian $\nabla^2 E$ was positive semi-definite. If a small perturbation is added to $\mathbf{v}^*$, then $\nabla^2 E$ becomes indefinite (see Theorem 2). In contrast, when $\zeta = -1.5$, $\nabla^2 E$ became *indefinite* (minimum eigenvalue $-0.2307$); this situation was similar to Fig.1(left).

**Remarks:** The eigen-spectrum (or curvature) variation along a line often arises in *separable* (i.e., mixed linear and nonlinear) optimization problems. As a small non-MLP model, consider, for instance, a separable objective function with $\boldsymbol{\theta} \equiv [p, v]^T$, two variables alone: $F(\boldsymbol{\theta}) = F(p, v) = pv^2$. Expressed below are the gradient and Hessian of $F$:

$$\nabla F = \begin{bmatrix} v^2 \\ 2pv \end{bmatrix}; \quad \nabla^2 F = \begin{bmatrix} 0 & 2v \\ 2v & 2p \end{bmatrix}. \tag{15}$$

Consider a line $v = 0$, where the Hessian $\nabla^2 F$ is *singular*. Then, the eigen-spectrum of $\nabla^2 F$ changes as the linear parameter $p$ alters while the first-order necessary condition ($\nabla F = \mathbf{0}$) is maintained with the objective-function value $F = 0$ held fixed. Clearly, $\nabla^2 F$ is positive semi-definite when $p > 0$, whereas it is negative semi-definite when $p < 0$. Hence, the line is a collection of degenerate stationary points. In this way, singularities may be closely related to flat regions, where any updates of

parameters do not change the objective function value. Back to MLP-learning, Blum [10] describes a different linear manifold of stationary points (see Sec.3.2 for more details), where the $\zeta$-adjusting procedure described above fails because $D=0$ (see **Example 3** below also). Some other types of linear manifolds (and eigen-spectrum changes) can be found; e.g., in [11, 4, 3]; unlike their work, *our paper did not claim anything about local minima,* and our approach is totally different.

**Example 3**: A linear-output five-weight 1-1-2-1 MLP with $\boldsymbol{\theta}=[p_1,p_2|v_1,v_2|w_1]^T$ (no thresholds), having tanh-hidden-node functions. If $\boldsymbol{\theta}^*=[1,1,0,0,0]^T$, then $\nabla E(\boldsymbol{\theta}^*)=\mathbf{0}$ with the *indefinite* Hessian $\nabla^2 E$ (hence, $\boldsymbol{\theta}^*$ a saddle point) below, in which all diagonal entries of $\mathbf{S}$ are zero due to $D=0$:

$$\nabla^2 E(\boldsymbol{\theta}^*) = \left[\begin{array}{cc|cc|c} 0 & 0 & 0 & 0 & 0 \\ 0 & 0 & 0 & 0 & 0 \\ \hline 0 & 0 & 0 & 0 & \star \\ 0 & 0 & 0 & 0 & \star \\ \hline 0 & 0 & \star & \star & 0 \end{array}\right] \quad \text{with} \quad \star \equiv \sum_{d=1}^{m} x_d r_d.$$

Here, $\star$ denotes a non-zero entry with input $x$ and residual $r$ over an arbitrary number $m$ of data. □

The point to note here is that it is important to look at the entire Hessian $\nabla^2 E$ of size $n \times n$. When $\mathbf{H}=\mathbf{O}$, a $4 \times 4$ block of zeros, $\nabla^2 E$ would be indefinite (again by the interlace theorem) as long as non-zero off-diagonal entries exist in $\nabla^2 E$, as in **Example 3** above. Needless to say, however, the Hessian analysis fails in certain pathological cases (see Sec.3.2). Typical is an aforementioned weight-$\infty$ case, where the sigmoid-shaped hidden-node functions are driven to saturation limits due to very large hidden weights. Then, only part of $\mathbf{J}^T\mathbf{J}$ associated with linear weights $\mathbf{p}$ appear in $\nabla^2 E$ since $\mathbf{S}=\mathbf{O}$ *even if residuals are still large.* This case is outside the scope of our analysis. It should be noted that a regularization scheme to penalize large weights is quite orthogonal to our scheme to exploit negative curvature. If a regularization term $\mu\boldsymbol{\theta}^T\boldsymbol{\theta}$ (with non-negative scalar $\mu$) is added to $E$, then the negative-curvature information will be lost due to $\nabla^2 E + \mu\mathbf{I}$.

# 3 The 2-2-1 MLP-learning examples found in the literature

In this section, we consider learning with a 2-2-1 MLP having nine weights; then, Eq.(6) reduces to
$$\boldsymbol{\theta}^T \equiv [\mathbf{p}^T|\mathbf{v}^T] = [\mathbf{p}^T|\mathbf{v}_1^T|\mathbf{v}_2^T] = [p_0, p_1, p_2|v_{0,1}, v_{1,1}, v_{2,1}|v_{0,2}, v_{1,2}, v_{2,2}],$$
where $\mathbf{v}_j$ is a (hidden) weight vector connecting to the $j$th hidden node. Here, all weights are non-linear since both hidden and final outputs are produced by sigmoidal logistic function $\phi(x) \equiv \frac{1}{1+e^{-x}}$.

## 3.1 Insensitivity to the initial weights in the singular XOR problem

The world-renowned XOR problem (involving only four data of binary values: ON and off) with a standard nine-weight 2-2-1 MLP is inevitably a *singular* problem because the Gauss-Newton Hessian $\mathbf{J}^T\mathbf{J}$ in Eq.(4) is always *singular* (at most rank 4), whereas $\mathbf{S}$ tends to be of (nearly) full rank; so does $\nabla^2 E$ (cf. rank analysis in [12]). This implies that *singularity* in terms of $\mathbf{J}^T\mathbf{J}$ is everywhere in the posed neuro-manifolds. It is well-known (e.g., see [13]) that the origin ($\mathbf{p}=\mathbf{0}$ and $\mathbf{v}=\mathbf{0}$) is a singular saddle point, where $\nabla E=\mathbf{0}$ and $\nabla^2 E=\mathbf{J}^T\mathbf{J}$ with only one positive eigenvalue and eight zeros. An interesting observation is that there always exists a descending path to the solution from any initial point $\boldsymbol{\theta}_{\text{init}}$ as long as $\boldsymbol{\theta}_{\text{init}}$ is randomly generated in a small range; i.e., in the vicinity of the origin. That is, first go directly down towards the origin from $\boldsymbol{\theta}_{\text{init}}$, and then move in a descent direction of negative curvature so as to escape from that singular saddle point. In this way, the 2-2-1 MLP can develop insensitivity to initial weights, always solving the posed XOR problem.

## 3.2 Blum's linear manifold of stationary points

In the XOR problem, Blum [10] found a line of stationary points by adding constraints to $\boldsymbol{\theta}$ as
$$L_1 \equiv v_{0,1}=v_{0,2}, \quad w_1 \equiv v_{1,1}=v_{2,2}, \quad w_2 \equiv v_{1,2}=v_{2,1}, w \equiv p_1=p_2, \quad (\text{with } L \equiv p_0), \quad (16)$$
leading to a *weight-sharing* MLP of five weights: $\boldsymbol{\theta} \equiv [L, w, L_1, w_1, w_2]^T$ following the notations in [10]. Using four XOR data: $(x_1, x_2; t) = \{(0,0; \text{off}), (0,1; \text{ON}), (1,0; \text{ON}), (1,1; \text{off})\}$ for $E$ in Eq.(3), Blum considered a point with $\mathbf{v}=\mathbf{0}$; hence, $\boldsymbol{\theta}^* \equiv [L, w, 0, 0, 0]^T$, which gives two identical hidden-node outputs: $h_1=h_2=\phi(0)=\frac{1}{1+e^0}=\frac{1}{2}$. This is the same situation as in Sec.2.2 and 2.3. By the constraints given in Eq.(16), the terminal output is given by $y=\phi(L+w)$. All those node outputs are independent of input data. Then, for a given target value "off" (e.g., 0.1), set
$$\text{ON} = 2\phi(L+w) - \text{off} \quad \Longleftrightarrow \quad \phi(L+w) = (\text{off} + \text{ON})/2 \quad (17)$$
so that those target values "off" and "ON" must approximate XOR.

**Blum's Claim** (page 539 [10]): *There are many stationary points that are not absolute minima. They correspond to $w$ and $L$ satisfying Eq.(17). Hence, they lie on a* **line** *"$L + w = c$ (constant)" in the $(w, L)$-plane. Actually, these points are local minima of $E$, being $\frac{1}{2}(\text{ON} - \text{off})^2$.* $\square$

A little algebra confirms that $\nabla E = \mathbf{0}$, and the quantities corresponding to $e$ and $D$ in Eq.(11) are all zeros; hence, $\mathbf{S} = \mathbf{O}$. Consequently, no matter how $\zeta$ (see Theorem 5) is changed to update $w$ and $L$ (along the line), $\nabla^2 E$ stays positive semi-definite, and $E$ in Eq.(3) remains the same 0.5 (flat region). This is certainly a limitation of the second-order Hessian analysis, and thus more efforts using higher-order derivatives were needed to disprove Blum's claim (see [14, 15]), and it turned out that Blum's line is a collection of *singular saddle* points. In what follows, we show what conditions must hold for the Hessian argument to work.

The 5-by-5 Hessian matrix $\nabla^2 E$ at a stationary point $\boldsymbol{\theta}^* = [L, w, 0, 0, 0]$ is given by

$$\underbrace{\nabla^2 E}_{5 \times 5} = \begin{bmatrix} 4A & 4A & 2wA & wA & wA \\ 4A & 4A & 2wA & wA & wA \\ 2wA & 2wA & w^2 A & \frac{w^2}{2}A & \frac{w^2}{2}A \\ wA & wA & \frac{w^2}{2}A & \frac{w^2}{8}(3A+S) & \frac{w^2}{8}(3A+S) \\ wA & wA & \frac{w^2}{2}A & \frac{w^2}{8}(3A+S) & \frac{w^2}{8}(3A+S) \end{bmatrix} \text{ with } \begin{cases} A \equiv \{\phi'(L+w)\}^2 \\ \\ S \equiv \phi''(L+w)\Big(\phi(L+w) - \text{off}\Big). \end{cases} \tag{18}$$

We thus obtain two non-zero eigenvalues of $\nabla^2 E$, $\lambda_1$ and $\lambda_2$, below using $k \equiv \frac{w^2}{8}(3A + S)$:

$$\lambda_1, \lambda_2 = \frac{1}{2} \left\{ [A(w^2 + 8) + 2k] \pm \sqrt{[A(w^2 + 8) + 2k]^2 - 2A(w^2 + 8)(4k - w^2 A)} \right\}. \tag{19}$$

Now, the smaller eigenvalue can be rendered *negative* when the following condition holds:

$$4k - w^2 A < 0 \iff A + S = \{\phi'(L+w)\}^2 + \phi''(L+w)\Big(\phi(L+w) - \text{off}\Big) < 0. \tag{20}$$

Choosing $L+w = 2$ and off $= 0.1$ accomplishes our goal, yielding *sign-different* eigenvalues with $\text{ON} = 2\phi(2) - \text{off} \approx 1.6616$ by Eq.(17). Because $\nabla^2 E$ is indefinite, the posed stationary point is a saddle point with $E = \frac{1}{2}(\text{ON} - \text{off})^2$ ($\approx 1.219$), as desired. In other words, the target value for ON is modified to break *symmetry* in data. Such a large target value ON (as 1.6616) is certainly un-attainable outside the range $(0,1)$ of the sigmoidal logistic function $\phi(x)$, but notice that ON is often set equal to 1.0, which is also un-attainable for finite weight values. It appears that the choice of such a (fictitiously large) value ON does not violate any Blum's assumption. When $0 \le \text{off} < \text{ON} \le 1$ (with $w \ne 0$), the Hessian $\nabla^2 E$ in Eq.(18) is always positive semi-definite of rank 2. Hence, it is a singular saddle point.

### 3.3 Two-class pattern classification problems of Gori and Tesi

We next consider two two-class pattern classification problems made by Gori & Tesi: one with five binary data (p.80 in [17]), and another with only three data (p.93 in [16]); see Fig.3. Both are *singular* problems, because $\text{rank}(\mathbf{J}^T\mathbf{J}) \le 5$; yet, both $\mathbf{S}$ and $\nabla^2 E$ tend to be of full rank; therefore, the $9 \times 9$ Hessian $\nabla^2 E$ tends to be *indefinite* (see Theorems 1 and 2). On p.81 in [17], a configuration of two separation lines, like two solid lines given by $\boldsymbol{\theta}_{\text{init}}$ in Fig.3(left) and (right), is claimed as a region attractive to a local-minimum point. Indeed, the batch-mode steepest-descent method fails to change the orientation of those solid lines. But its failure does not imply that there is no *descending* way out of the two-solid-line configuration given by $\boldsymbol{\theta}_{\text{init}}$ because the convergence of the steepest-descent method to a (local) minimizer can be guaranteed by examining *negative curvature* (e.g., p.45 in [18]). We shall show a descending negative curvature direction.

In the five-data case, the steepest-descent method moves $\boldsymbol{\theta}_{\text{init}}$ to a point, where the weights become relatively large; the gradient vector $\nabla E \approx \mathbf{0}$; the Hessian $\nabla^2 E$ is positive semi-definite; and Eq.(3) with $m = 5$ is given by $E = \frac{1}{3}(\text{ON} - \text{off})^2$, for which the two residuals at data points (0,0) and (1,1) are made zeros. We can find such a point analytically by a linear-equation solving: Given $\boldsymbol{\theta}_{\text{init}}$ in Fig.3, the solution to the linear system below yields $\mathbf{p}^* = [p_0^*, p_1^*, p_2^*]^T$ (three terminal weights):

$$\begin{bmatrix} 1 & \phi(-1.5) & \phi(-0.5) \\ 1 & \phi(0.5) & \phi(1.5) \\ 1 & \phi(-0.5) & \phi(0.5) \end{bmatrix} \begin{bmatrix} p_0^* \\ p_1^* \\ p_2^* \end{bmatrix} = \begin{bmatrix} \phi^{-1}(\text{off}) \\ \phi^{-1}(\text{off}) \\ \phi^{-1}\left(\frac{2\text{ON}+\text{off}}{3}\right) \end{bmatrix}.$$

The resulting point $\boldsymbol{\theta}^* \equiv [p_0^*, p_1^*, p_2^*; -1.5, 1, 1; -0.5, 1, 1]^T$, where the norm of $\mathbf{p}^*$ becomes relatively large $O(10^2)$, gives the zero gradient vector, the positive semi-definite Hessian of rank 5, and $E = \frac{1}{3}(\text{ON} - \text{off})^2$, as mentioned above. It is observed, however, that small perturbations on $\boldsymbol{\theta}^*$ render

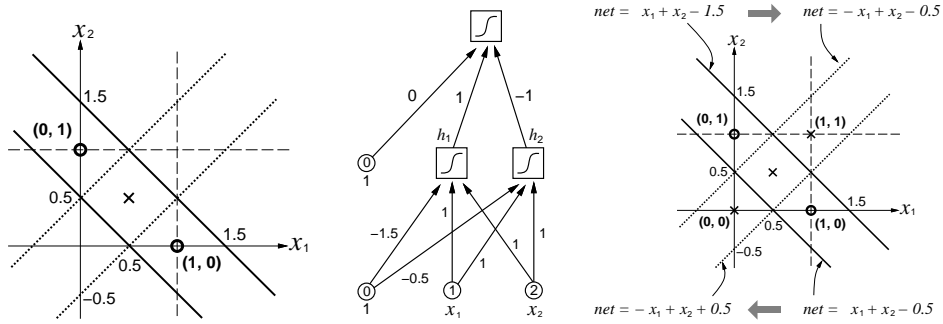

Figure 3: Gori & Tesi's two-class pattern classification problems (left) three-data case; (right) five-data case; and (middle) a 2-2-1 MLP with initial weight values $\boldsymbol{\theta}_{\text{init}} \equiv [0, 1, -1; -1.5, 1, 1; -0.5, 1, 1]^T$. Its corresponding initial configuration gives two solid lines of net-inputs (to two hidden nodes) in the input space, where "○" stands for two ON-data (1,0), (0,1), whereas "×" for one off-data (0.5,0.5) in left figure and three off-data (0,0), (0.5,0.5), (1,1) in right figure. A solution to both problems may be given by the two dotted lines with $\boldsymbol{\theta}_{\text{sol}} \equiv [0, 1, -1; -0.5, -1, 1; 0.5, -1, 1]^T$.

$\nabla^2 E$ *indefinite* of full rank (since **S** is dominant): $\text{rank}(\mathbf{S}) = \text{rank}(\nabla^2 E) = 9$ with $\text{rank}(\mathbf{J}^T\mathbf{J}) = 4$; this suggests a descend direction (other than the steepest descent) to follow from $\boldsymbol{\theta}_{\text{init}}$ to a solution $\boldsymbol{\theta}_{\text{sol}}$. Fig.3(right) presents one of them, an intuitive change of six hidden weights (with the other three weights held fixed) from two solid lines to two dotted ones, indicated by two thick arrows given by $\Delta\boldsymbol{\theta} \equiv \boldsymbol{\theta}_{\text{sol}} - \boldsymbol{\theta}_{\text{init}} = [0, 0, 0; 1, -2, 0; 1, -2, 0]^T$, is a *descent* direction of negative curvature down to $\boldsymbol{\theta}_{\text{sol}}$ because $\Delta\boldsymbol{\theta}^T\nabla^2 E(\boldsymbol{\theta}_{\text{init}})\Delta\boldsymbol{\theta} < 0$, where $\nabla^2 E(\boldsymbol{\theta}_{\text{init}})$, the Hessian evaluated at $\boldsymbol{\theta}_{\text{init}}$, was indefinite. Intriguingly enough, it is easy to confirm for the three-data case that the posed "descent" direction of negative curvature $\Delta\boldsymbol{\theta}$ is *orthogonal* to $-\nabla E$, the steepest-descent direction.

**Claim**: Line search from $\boldsymbol{\theta}_{\text{init}}$ to $\boldsymbol{\theta}_{\text{sol}}$ *monotonically decreases* the squared error $E(\boldsymbol{\theta}_{\text{init}} + \eta\Delta\boldsymbol{\theta})$ as the step size $\eta$ (scalar) changes from 0 to 1; hence, *no plateau*.

**Proof** for the three-data case: (The five-data case can be proved in a similar fashion.) Using target values ON=1 and off=0, let $q(\eta) \equiv E(\boldsymbol{\theta}_{\text{init}} + \eta\Delta\boldsymbol{\theta}) - E(\boldsymbol{\theta}_{\text{init}})$. Then, we show below that $q'(\eta) < 0$ using a property that $\phi(-x) = 1 - \phi(x)$:

$q(\eta) = \frac{1}{2}\{\phi(-0.5-\eta) - \phi(0.5-\eta) - \text{ON}\}^2 + \frac{1}{2}\{\phi(-0.5+\eta) - \phi(0.5+\eta) - \text{ON}\}^2 - \{\phi(-0.5) - \phi(0.5) - \text{ON}\}^2$

$= \frac{1}{2}\{1 - \phi(0.5+\eta) - \phi(0.5-\eta) - \text{ON}\}^2 + \frac{1}{2}\{1 - \phi(0.5-\eta) - \phi(0.5+\eta) - \text{ON}\}^2 - \{1 - \phi(0.5) - \phi(0.5) - \text{ON}\}^2$

$= \{1 - \text{ON} - \phi(0.5+\eta) - \phi(0.5-\eta)\}^2 - \{1 - \text{ON} - 2\phi(0.5)\}^2$

$= \{\phi(0.5+\eta) + \phi(0.5-\eta)\}^2 - 4\{\phi(0.5)\}^2$.

Differentiation leads to $q'(\eta) = 2\{\phi(0.5+\eta) + \phi(0.5-\eta)\}\{\phi'(0.5+\eta) - \phi'(0.5-\eta)\} < 0$ because $\phi(0.5+\eta) > 0$, $\phi(0.5-\eta) > 0$, and $\eta > 0$, which guarantees $\phi'(0.5+\eta) < \phi'(0.5-\eta)$. □

# 4 Summary

In a general setting, we have proved that negative curvature can arise in MLP-learning. To make it analytically tractable, we intentionally used *noise-free* small data sets but on "noisy" data, the conditions for Theorems 1 and 2 most likely hold in the vicinity of singularity regions; it then follows that the Hessian $\nabla^2 E$ tends to be indefinite (of nearly full rank). Our numerical results confirm that the negative-curvature information is of immense value for escaping from singularity plateaus including some problems where no method was developed to alleviate plateaus. In simulation, we employed the second-order stagewise backpropagation [12] (that can evaluate $\nabla^2 E$ and $\mathbf{J}^T\mathbf{J}$ at the essentially same cost; see proof therein) to obtain $\nabla^2 E$ explicitly and its eigen-directions so as to exploit negative curvature. This approach is suitable for up to medium-scale problems, for which our analysis suggests using existing trust-region globalization strategies whose theory has thrived on negative curvature including indefinite dogleg [19]. For large-scale problems, one could resort to matrix-free Krylov subspace methods: Among them, the truncated conjugate-gradient (Krylov-dogleg) method tends to pick up an arbitrary negative curvature (hence, slowing down learning; see [20] for numerical evidence); so, other trust-region Krylov subspace methods are of our great interest such as a Lanczos type [21] and a parameterized eigenvalue approach [22].

**Acknowledgments**

The work is partially supported by the National Science Council, Taiwan (NSC-99-2221-E-011-097).

# References

[1] Amari, S.-I., Park,H. & Fukumizu, K. Adaptive Method of Realizing Natural Gradient Learning for Multilayer Perceptrons. *Neural Computation*, 12:1399-1409, 2000.

[2] Amari, S.-I., Park, H. & Ozeki, T. Singularities affect dynamics of learning in neuro-manifolds. *Neural Computation*, 18(5):1007-1065, 2006.

[3] Wei, H., Zhang, J., Cousseau, F., Ozeki, T., & Amari, S.-I. Dynamics of Learning Near Singularities in Layered Networks. *Neural Computation*, 20(3):813-843, 2008.

[4] Fukumizu, K. & Amari, S.-I. Local Minima and Plateaus in Hierarchical Structures of Multilayer Perceptrons. *Neural Networks*, 13(3):317–327, 2000.

[5] Jennrich, R.I. & Sampson, P.F. Application of Stepwise Regression to Non-Linear Estimation. *Technometrics*, 10(1):63–72, 1968.

[6] Cousseau, F., Ozeki, T., & Amari, S.-I. Dynamics of Learning in Multilayer Perceptrons near Singularities *IEEE Trans. on Neural Networks*, 19(8):1313-1328, 2008.

[7] Powell, M.J.D. A hybrid method for nonlinear equations. In *Numerical Methods for Nonlinear Algebraic Equations*, Ed. by P.Rabinowitz, Gordon & Breach, London, pp.87–114, 1970.

[8] Dennis, J.E., Jr. Nonlinear least squares and equations. In *The state of the art in numerical analysis*, Ed. by D. Jacobs, Academic Press, London, pp.269–312, 1977.

[9] Parlett, B.N. The Symmetric Eigenvalue Problem. *SIAM*, 1998.

[10] Blum, E.K. Approximation of Boolean Functions by Sigmoidal Networks: Part I: XOR and other two-variable functions. *Neural Computation*, 1:532-540, 1989.

[11] Sprinkhuizen-Kuyper, I.G. & Boers, E.J.W. A Local Minimum for the 2-3-1 XOR Network. *IEEE Transactions on Neural Networks*, 10(4):968–971, 1999.

[12] Mizutani, E. & Dreyfus, S.E. Second-order stagewise backpropagation for Hessian-matrix analyses and investigation of negative curvature. *Neural Networks*, vol.21 (issues 2–3):193-203, 2008. (See its Corrigendum in vol.21, issue 9, page 1418).

[13] Sprinkhuizen-Kuyper, I.G. & Boers, E.J.W. The error surface of the 2-2-1 XOR network: The finite stationary points. *Neural Networks*, 11:683–690, 1998.

[14] Tsaih, R.-H. An Improved Back Propagation Neural Network Learning Algorithm. Ph.D thesis at the Department of Industrial Engineering and Operations Research, University of California at Berkeley, pp.67–70, 1991.

[15] Sprinkhuizen-Kuyper, I.G. & Boers, E.J.W. A comment on a paper of Blum: Blum's local minima are saddle points. *Tech. Rep. No. 94-34,* Leiden University, Department of Computer Science, Leiden, The Netherlands, 1994.

[16] Gori, M. & Tesi, A. Some examples of local minima during learning with backpropagation. *Third Italian Workshop on Parallel Architectures and Neural Networks.* (Ed. by E.R. Caianiello), World Scientific Publishing Co., pp. 87–94, 1990.

[17] Gori, M. & Tesi, A. On the Problem of Local Minima in Backpropagation. *IEEE Trans. on Pattern Analysis and Machine Intelligence*, 14(1):76-86, 1992.

[18] Nocedal, J & Wright, S.J. *Numerical Optimization.* Springer Verlag, 1999.

[19] Byrd, R.H., Schnabel, R.B. & Schultz, G.A. Approximate solution of the trust region problems by minimization over two-dimensional subspaces. *Mathematical Programming*, 40:247–263, 1988.

[20] Mizutani, E. & Demmel, J.W. Iterative scaled trust-region learning in Krylov subspaces via Pearlmutter's implicit sparse Hessian-vector multiply. In S. Thrun, L. Saul, and B. Schölkopf, editors, *Advances in Neural Information Processing Systems,* MIT Press, 16:209–216, 2004.

[21] Gould, N.I.M., Lucidi, S., Roma, M. & Toint, Ph.L. Solving the trust-region subproblem using the Lanczos method. *SIAM Journal on Optimization*, 9(2):504–525, 1999.

[22] Rojas, M., Santos, S.A. & Sorensen, D.C. A New Matrix-Free Algorithm for the Large-Scale Trust-Region Subproblem. *SIAM Journal on Optimization*, 11(3):611–646, 2000.

